# Generalization Performance of Some Learning Problems in Hilbert Functional Spaces

**Tong Zhang**
IBM T.J. Watson Research Center
Yorktown Heights, NY 10598
tzhang@watson.ibm.com

## Abstract

We investigate the generalization performance of some learning problems in Hilbert functional Spaces. We introduce a notion of convergence of the estimated functional predictor to the best underlying predictor, and obtain an estimate on the rate of the convergence. This estimate allows us to derive generalization bounds on some learning formulations.

## 1 Introduction

In machine learning, our goal is often to predict an unobserved output value $y$ based on an observed input vector $x$. This requires us to estimate a functional relationship $y \approx p(x)$ from a set of example pairs of $(x, y)$. Usually the quality of the predictor $p(x)$ can be measured by a loss function $L(p(x), y)$ that is problem dependent. In machine learning, we assume that the data $(x, y)$ are drawn from an underlying distribution $D$ which is not known. Our goal is to find $p(x)$ so that the expected true loss of $p$ given below is as small as possible:

$$R(p(\cdot)) = E_{(x,y) \sim D} L(p(x), y).$$

In order to estimate a good predictor $p(x)$ from a set of training data $(x, y)$ randomly drawn from $D$, it is necessary to start with a model of the functional relationship. In this paper, we consider models that are subsets in some Hilbert functional spaces $H$. Denote by $\| \cdot \|_H$ the norm in $H$, we consider models in the set $\Omega = \{p(x) \in H : \|p(x)\|_H \leq A\}$, where $A$ is a parameter that can be used to control the size of the underlying model family. We would like to find the best model in $\Omega$ which is given by:

$$\hat{p}(\cdot) = \arg \min_{p \in \Omega} E_{(x,y)} L(p(x), y). \tag{1}$$

By introducing a non-negative Lagrangian multiplier $\lambda \geq 0$, we may rewrite the above problem as:

$$\hat{p}(\cdot) = \arg \min_{p \in H} [E_{(x,y)} L(p(x), y) + \frac{\lambda}{2} \|p(\cdot)\|_H^2]. \tag{2}$$

We shall only consider this equivalent formulation in this paper. In addition, for technical reasons, we also assume that $L(a, b)$ is a convex function of $a$.

Given $n$ training examples $(x_1, y_1), \dots, (x_n, y_n)$, we consider the following estimation method to approximate the optimal predictor $\hat{p}(x) \in \Omega$:

$$\hat{p}_n(\cdot) = \arg\min_{p \in H}\left[\frac{1}{n}\sum_{i=1}^{n} L(p(x_i), y_i) + \frac{\lambda}{2}\|p(\cdot)\|_H^2\right]. \tag{3}$$

The goal of this paper is to show that as $n \to \infty$, $\hat{p}_n \to \hat{p}$ in probability under appropriate regularity conditions. Furthermore, we obtain an estimate on the rate of convergence. Consequences of this result in some specific learning formulations are examined.

## 2   Convergence of the estimated predictor

Assume that input $x$ belongs to a set $X$. We make the reasonable assumption that $p$ is point-wise continuous under the $\|\cdot\|_H$ topology: $\forall x \in X, \lim_{p \to p_0} p(x) = p_0(x)$ where $p \to p_0$ is in the sense that $\|p - p_0\|_H \to 0$. This assumption is equivalent to $\sup_{\|p\|_H \le 1} p(x) < +\infty \;\forall x \in X$. The condition implies that each data point $x$ can be regarded as a bounded linear functional $q_x$ on $H$ such that $\forall p \in H$: $q_x(p) = p(x)$. Since a Hilbert space $H$ is self-dual, we can represent $q_x$ by an element in $H$. For notational simplicity, $\forall x$ we shall let $q_x \in H$ defined as $q_x \cdot p = p(x)$ for all $p \in H$, where $\cdot$ denotes the inner product of $H$.

It is clear that $q_x$ can be regarded as a representing feature vector of $x$ in $H$. This representation can be computed as follows. Let $\tilde{p} = \arg\max_{\|p\|_H = 1} p(x)$, then it is not difficult to see that $\tilde{p} = q_x/\|q_x\|_H$ and $\tilde{p}(x) = \|q_x\|_H$. It follows that $q_x(\cdot) = \tilde{p}(x)\tilde{p}(\cdot)$. Note that this method of computing $q_x$ is not important for the purpose of this paper.

Since $p(x) = p \cdot q_x$ can now be considered as a linear functional using the feature space representation $q_x$ of $x$, we can use the idea from [6] to analyze the convergence behavior of $\hat{p}_n$ in $H$. Following [6], using the linear representation of $p(x)$, we differentiate (2) at the optimal solution $\hat{p}(\cdot)$, which leads to the following first order condition:

$$E_{x,y}L_1'(\hat{p} \cdot q_x, y)q_x + \lambda\hat{p} = 0, \tag{4}$$

where $L_1'(a, b)$ is the derivative of $L(a, b)$ with respect to $a$ if $L$ is smooth; it denotes a subgradient (see [4]) otherwise. Since we have assumed that $L(a, b)$ is a convex function of $a$, we know that $L(a_1, b) + (a_2 - a_1)L_1'(a_1, b) \le L(a_2, b)$. This implies the following inequality:

$$\frac{1}{n}\sum_{i=1}^{n} L(\hat{p}(x_i), y_i) + \frac{1}{n}\sum_{i=1}^{n} L_1'(\hat{p}(x_i), y_i)(\hat{p}_n(x_i) - \hat{p}(x_i)) \le \frac{1}{n}\sum_{i=1}^{n} L(\hat{p}_n(x_i), y_i),$$

which is equivalent to:

$$\frac{1}{n}\sum_{i=1}^{n} L(\hat{p}(x_i), y_i) + \frac{\lambda}{2}\hat{p}^2 +$$

$$\left[\frac{1}{n}\sum_{i=1}^{n} L_1'(\hat{p}(x_i), y_i)(\hat{p}_n(x_i) - \hat{p}(x_i)) + \lambda\hat{p} \cdot (\hat{p}_n - \hat{p})\right] + \frac{\lambda}{2}(\hat{p} - \hat{p}_n)^2$$

$$\le \frac{1}{n}\sum_{i=1}^{n} L(\hat{p}_n(x_i), y_i) + \frac{\lambda}{2}\hat{p}_n^2.$$

Note that we have used $p^2$ to denote $p \cdot p = \|p\|_H^2$. Also note that by the definition of $\hat{p}_n$, we have

$$\frac{1}{n}\sum_{i=1}^{n} L(\hat{p}(x_i), y_i) + \frac{\lambda}{2}\hat{p}^2 \ge \frac{1}{n}\sum_{i=1}^{n} L(\hat{p}_n(x_i), y_i) + \frac{\lambda}{2}\hat{p}_n^2.$$

Therefore by comparing the above two inequalities, we obtain:

$$\frac{\lambda}{2}(\hat{p} - \hat{p}_n)^2 \leq -[\frac{1}{n}\sum_{i=1}^{n} L_1'(\hat{p}(x_i), y_i)q_{x_i} \cdot (\hat{p}_n - \hat{p}) + \lambda\hat{p} \cdot (\hat{p}_n - \hat{p})]$$

$$\leq \|\frac{1}{n}\sum_{i=1}^{n} L_1'(\hat{p}(x_i), y_i)q_{x_i} + \lambda\hat{p}\|_H \|\hat{p} - \hat{p}_n\|_H.$$

This implies that

$$\|\hat{p} - \hat{p}_n\|_H \leq \frac{2}{\lambda}\|\frac{1}{n}\sum_{i=1}^{n} L_1'(\hat{p}(x_i), y_i)q_{x_i} + \lambda\hat{p}\|_H$$

$$= \frac{2}{\lambda}\|\frac{1}{n}\sum_{i=1}^{n} L_1'(\hat{p}(x_i), y_i)q_{x_i} - E_{x,y}L_1'(\hat{p}(x), y)q_x\|_H. \tag{5}$$

Note that the last equality follows from the first order condition (4). This is the only place the condition is used. In (5), we have already bounded the convergence of $\hat{p}_n$ to $\hat{p}$ in terms of the convergence of the empirical expectation of a random vector $L_1'(\hat{p}(x), y)q_x$ to its mean. The latter is often easier to estimate. For example, if its variance can be bounded, then we may use the Chebyshev inequality to obtain a probability bound. In this paper, we are interested in obtaining an exponential probability bound. In order to do so, similar to the analysis in [6], we use the following form of concentration inequality which can be found in [5], page 95:

**Theorem 2.1 ([5])** *Let $\xi_i$ be zero-mean independent random vectors in a Hilbert space $H$. If there exists $M > 0$ such that for all natural numbers $l \geq 2$: $\sum_{i=1}^{n} E\|\xi_i\|_H^l \leq \frac{nb}{2}l!M^l$. Then for all $\delta > 0$: $P(\|\frac{1}{n}\sum_i \xi_i\|_H \geq \delta) \leq 2\exp(-\frac{n}{2}\delta^2/(bM^2 + \delta M))$.*

We may now use the following form of Jensen's inequality to bound the moments of the zero-mean random vector $L_1'(\hat{p}(x_i), y_i)q_{x_i} - E_{x,y}L_1'(\hat{p}(x), y)q_x$:

$$E|\xi - E\xi|^l \leq 2^{l-1}(E|\xi|^l + |E\xi|^l) \leq 2^{l-1}(E|\xi|^l + E|\xi|^l) = 2^l E|\xi|^l.$$

From inequality (5) and Theorem 2.1, we immediately obtain the following bound:

**Theorem 2.2** *If there exists $M > 0$ such that for all natural numbers $l \geq 2$: $E_{x,y}|L_1'(\hat{p}(x), y)\|q_x\|_H|^l \leq \frac{b}{2}l!M^l$. Then for all $\delta > 0$:*

$$P(\|\hat{p}_n - \hat{p}\|_H \geq \delta) \leq 2\exp(-\frac{n}{8}\lambda^2\delta^2/(4bM^2 + \lambda\delta M)).$$

Although Theorem 2.2 is quite general, the quantity $b$ and $M$ on the right hand side of the bound depend on the optimal predictor $\hat{p}$ which requires to be estimated. In order to obtain a bound that does not require any knowledge of the true distribution $D$, we may impose the following assumptions: both $L_1'(a, b)$ and $\|q_x\|_H$ are bounded. Observe that $\|q_x\|_H = \sup_{p \in H: \|p\|_H=1} p(x)$, we obtain the following result:

**Corollary 2.1** *Assume that $\forall x \in X$, $\sup_{p \in H: \|p\|_H=1} p(x) \leq \kappa$. Also assume that the loss function $L(a, b)$ satisfies the condition that $|L_1'(a, b)| \leq M$, then $\forall \delta > 0$:*

$$P(\|\hat{p}_n - \hat{p}\|_H \geq \delta) \leq 2\exp(-\frac{n}{8}\lambda^2\delta^2/(4\kappa^2M^2 + \kappa M\lambda\delta)).$$

# 3 Generalization performance

We study some consequences of Corollary 2.1, which bounds the convergence rate of the estimated predictor to the best predictor.

## 3.1 Regression

We consider the following type of Huber's robust loss function:

$$L(p, y) = \begin{cases} \frac{1}{2}(p - y)^2 & \text{if } |p - y| \leq M \\ M|p - y| - \frac{M^2}{2} & \text{if } |p - y| \geq M. \end{cases} \quad (6)$$

It is clear that $L$ is continuous differentiable and $|L_1'(p, y)| \leq M$ for all $p$ and $y$. It is also not hard to check that $\forall p_1, p_2$ and $y$:

$$L(p_2, y) - L(p_1, y) - (p_2 - p_1)L_1'(p_1, y) \leq \frac{1}{2}(p_2 - p_1)^2.$$

Using this inequality and (4), we obtain:

$$[E_{x,y}L(\hat{p}_n(x), y) + \frac{\lambda}{2}\|\hat{p}_n\|_H^2] - [E_{x,y}L(\hat{p}(x), y) + \frac{\lambda}{2}\|\hat{p}\|_H^2]$$

$$= E_{x,y}(L(\hat{p}_n(x), y) - L(\hat{p}(x), y) - L_1'(\hat{p}(x), y)(\hat{p}_n(x) - \hat{p}(x))) + \frac{\lambda}{2}\|\hat{p}_n - \hat{p}\|_H^2$$

$$\leq E_{x,y}\frac{1}{2}(\hat{p}_n(x) - \hat{p}(x))^2 + \frac{\lambda}{2}\|\hat{p}_n - \hat{p}\|_H^2.$$

If we assume that $\|\hat{p}_n - \hat{p}\|_H \leq \delta$ and $\sup_{p \in H : \|p\|_H = 1} p(x) \leq \kappa$, then

$$E_{x,y}L(\hat{p}_n(x), y) \leq E_{x,y}L(\hat{p}(x), y) + \frac{\lambda}{2}(\|\hat{p}\|_H^2 - \|\hat{p}_n\|_H^2) + \frac{\delta^2}{2}(\kappa^2 + \lambda). \quad (7)$$

This gives the following inequality:

$$E_{x,y}L(\hat{p}_n(x), y) - E_{x,y}L(\hat{p}(x), y) \leq \lambda\delta\|\hat{p}_n\|_H + \delta^2(\frac{\kappa^2}{2} + \lambda).$$

It is clear that the right-hand side of the above inequality does not depend on the unobserved function $\hat{p}$. Using Corollary 2.1, we obtain the following bound:

**Theorem 3.1** *Using loss function (6) in (3). Assume that $\sup_{p \in H : \|p\|_H = 1} p(x) \leq \kappa$, then $\forall 0 < \delta \leq \kappa M/\lambda$, with probability of at least $1 - 2\exp(-n\lambda^2\delta^2/40\kappa^2 M^2)$, we have*

$$E_{x,y}L(\hat{p}_n(x), y) \leq E_{x,y}L(\hat{p}(x), y) + \lambda\delta\|\hat{p}_n\|_H + \delta^2(\frac{\kappa^2}{2} + \lambda).$$

Theorem 3.1 compares the performance of the computed function with that of the optimal predictor $\hat{p} \in \Omega \subset H$ in (1). This style of analysis has been extensively used in the literature. For example, see [3] and references therein. In order to compare with their results, we can rewrite Theorem 3.1 in another form as: with probability of at least $1 - \eta$,

$$E_{x,y}L(\hat{p}_n(x), y) \leq E_{x,y}L(\hat{p}(x), y) + O(\sqrt{\frac{1 - \ln\eta}{n}}).$$

In [3], the authors employed a covering number analysis which led to a bound of the form (for squared loss)

$$E_{x,y}L(\hat{p}_n(x), y) \leq E_{x,y}L(\hat{p}(x), y) + O(\frac{\ln n - \ln\eta}{n})$$

for finite dimensional problems. Note that the constant in their $O(\cdot)$ depends on the pseudo-dimension, which can be infinity for problems considered in this paper. It is possible to employ their analysis using some covering number bounds for general Hilbert spaces. However, such an analysis would have led to a result of the following form for our problems:

$$E_{x,y}L(\hat{p}_n(x), y) \leq E_{x,y}L(\hat{p}(x), y) + O((\frac{\ln n - \ln \eta}{n})^{1/3}).$$

It is also interesting to compare Theorem 3.1 with the leave-one-out analysis in [7]. The generalization error averaged over all training examples for squared loss can be bounded as

$$E_{x,y}L(\hat{p}_n(x), y) \leq (1 + O(\frac{1}{\lambda n}))(E_{x,y}L(\hat{p}(x), y) + \lambda\|\hat{p}\|_H^2).$$

This result is not directly comparable with Theorem 3.1 since the right hand side includes an extra term of $\lambda\|\hat{p}\|_H^2$. Using the analysis in this paper, we may obtain a similar result from (7) which leads to an average bound of the form:

$$E_{x,y}L(\hat{p}_n(x), y) \leq (E_{x,y}L(\hat{p}(x), y) + \lambda\|\hat{p}\|_H^2) + O(\frac{1}{\lambda^2 n}).$$

It is clear that the term $O(\frac{1}{\lambda^2 n})$ resulted in our paper is not as good as $O(\frac{1}{\lambda n})$ from [7]. However analysis in this paper leads to probability bounds while the leave-one-out analysis in [7] only gives average bounds. It is also worth mentioning that it is possible to refine the analysis presented in this section to obtain a probability bound which when averaged, gives a bound with the correct term of $O(\frac{1}{\lambda n})$, rather than $O(\frac{1}{\lambda n})$ in the current analysis. However due to the space limitation, we shall skip this more elaborated derivation.

In addition to the above style bounds, it is also interesting to compare the generalization performance of the computed function to the empirical error of the computed function. Such results have occurred, for example, in [1]. In order to obtain a comparable result, we may use a derivation similar to that of (7), together with the first order condition of (3) as follows:

$$\frac{1}{n}\sum_{i=1}^{n} L_1'(\hat{p}_n \cdot q_{x_i}, y_i)q_{x_i} + \lambda\hat{p}_n = 0.$$

This leads to a bound of the form:

$$\frac{1}{n}\sum_{i=1}^{n} L(\hat{p}(x_i), y_i) \leq \frac{1}{n}\sum_{i=1}^{n} L(\hat{p}_n(x_i), y_i) + \frac{\lambda}{2}(\|\hat{p}_n\|_H^2 - \|\hat{p}\|_H^2) + \frac{\delta^2}{2}(\kappa^2 + \lambda).$$

Combining the above inequality and (7), we obtain the following theorem:

**Theorem 3.2** *Using loss function (6) in (3). Assume that $\sup_{p\in H:\|p\|_H=1} p(x) \leq \kappa$, then $\forall 0 < \delta \leq \kappa M/\lambda$, with probability of at least $1 - 2\exp(-n\lambda^2\delta^2/40\kappa^2 M^2)$, we have*

$$E_{x,y}L(\hat{p}_n(x), y) - \frac{1}{n}\sum_{i=1}^{n} L(\hat{p}_n(x_i), y_i)$$

$$\leq \delta^2(\kappa^2 + \lambda) + [E_{x,y}L(\hat{p}(x), y) - \frac{1}{n}\sum_{i=1}^{n} L(\hat{p}(x_i), y_i)].$$

Unlike Theorem 3.1, the bound given in Theorem 3.2 contains a term $[E_{x,y}L(\hat{p}(x), y) - \frac{1}{n}\sum_{i=1}^{n} L(\hat{p}(x_i), y_i)]$ which relies on the unknown optimal predictor $\hat{p}$. From Theorem 3.1, we know that this term does not affect the performance of the estimated function $\hat{p}_n$ when

compared with the performance of $\hat{p}$. In order for us to compare with the bound in [1] obtained from an algorithmic stability point of view, we make the additional assumption that $|\hat{p}(x) - y| \leq M$ for all $(x, y)$. Note that this assumption is also required in [1]. Using Hoeffding's inequality, we obtain that with probability of at most $\exp(-n\lambda^2\delta^2/40\kappa^2 M^2)$,

$$E_{x,y}L(\hat{p}(x), y) - \frac{1}{n}\sum_{i=1}^{n} L(\hat{p}(x_i), y_i) > \lambda\delta M/(8\sqrt{5}\kappa).$$

Together with Theorem 3.2, we have with probability of at least $1 - \eta$,

$$E_{x,y}L(\hat{p}_n(x), y) - \frac{1}{n}\sum_{i=1}^{n} L(\hat{p}_n(x_i), y_i) \leq (\kappa^2 + \lambda)\frac{40\kappa^2 M^2 \ln\frac{3}{\eta}}{\lambda^2 n} + M^2\sqrt{\frac{\ln\frac{3}{\eta}}{8n}}.$$

This compares very favorably to the following bound in [1]:[1]

$$E_{x,y}L(\hat{p}_n(x), y) - \frac{1}{n}\sum_{i=1}^{n} L(\hat{p}_n(x_i), y_i) \leq \frac{2\kappa^2 M^2}{\lambda n} + 2M^2\sqrt{(\frac{2\kappa^4}{\lambda^2} + \frac{4\kappa^2}{\lambda} + 2)\frac{\ln\frac{2}{\eta}}{n}}.$$

### 3.2 Binary classification

In binary classification, the output value $y \in \{\pm 1\}$ is a discrete variable. Given a continuous model $p(x)$, we consider the following prediction rule: predict $y = 1$ if $p(x) \geq 0$, and predict $y = -1$ otherwise. The classification error (we shall ignore the point $p(x) = 0$, which is assumed to occur rarely) is

$$I(p(x), y) = \begin{cases} 1 & \text{if } p(x)y \leq 0, \\ 0 & \text{if } p(x)y > 0. \end{cases}$$

Unfortunately, this classification error function is not convex, which cannot be handled in our formulation. In fact, even in many other popular methods, such as logistic regression and support vector machines, some kind of convex formulations have to be employed. We shall thus consider the following soft-margin SVM style loss as an illustration:

$$L(p, y) = \max(1 - py, 0). \tag{8}$$

Note that the separable case of this loss was investigated in [6]. In this case, $L_1'(p, y)$ denotes a subgradient rather than gradient since $L(p, y)$ is non-smooth: at $py = 1$, $L_1'(p, y)y \in [-1, 0]$; $L_1'(p, y)y = -1$ when $py < 1$ and $L_1'(p, y) = 0$ when $py > 1$.

Since $\forall |p_1 - p_2| \leq \delta$ and $y = \pm 1$, $I(p_2, y) \leq I(p_1 - \delta y, y)$, we know that if $\|\hat{p}_n - \hat{p}\|_H \leq \delta$, then

$$E_{x,y}I(\hat{p}_n(x), y) \leq E_{x,y}I(\hat{p}(x) - \delta\kappa y, y),$$
$$\frac{1}{n}\sum_{i=1}^{n} I(\hat{p}(x_i) - \delta\kappa y_i, y_i) \leq \frac{1}{n}\sum_{i=1}^{n} I(\hat{p}_n(x_i) - 2\delta\kappa y_i, y_i).$$

Using the standard Hoeffding's inequality, we have with probability of at most $\exp(-n\lambda^2\delta^2/40\kappa^2)$,

$$E_{x,y}I(\hat{p}(x) - \delta\kappa y, y) > \frac{1}{n}\sum_{i=1}^{n} I(\hat{p}(x_i) - \delta\kappa y_i, y_i) + \lambda\delta/(4\sqrt{5}\kappa).$$

When $\frac{1}{n} \sum_{i=1}^{n} I(\hat{p}(x_i) - \delta\kappa y_i, y_i) \approx 0$, it is usually better to use a different (multiplicative) form of Hoeffding's inequality, which implies that with probability of at most $\exp(-n\lambda^2\delta^2/40\kappa^2)$,

$$E_{x,y}I(\hat{p}(x) - \delta\kappa y, y) > \max(\frac{2}{n}\sum_{i=1}^{n} I(\hat{p}(x_i) - \delta\kappa y_i, y_i), \lambda^2\delta^2/(5\kappa^2)).$$

Together with Corollary 2.1, we obtain the following margin-percentile result:

**Theorem 3.3** *Using loss function (8) in (3). Assume that* $\sup_{p \in H: \|p\|_H = 1} p(x) \leq \kappa$, *then* $\forall 0 < \delta \leq \kappa/\lambda$, *with probability of at least* $1 - 3\exp(-n\lambda^2\delta^2/40\kappa^2)$, *we have*

$$E_{x,y}I(\hat{p}_n(x), y) \leq \frac{1}{n}\sum_{i=1}^{n} I(\hat{p}_n(x_i) - 2\delta\kappa y_i, y_i) + \lambda\delta/(4\sqrt{5}\kappa).$$

*We also have with probability of at least* $1 - 3\exp(-n\lambda^2\delta^2/40\kappa^2)$,

$$E_{x,y}I(\hat{p}_n(x), y) \leq \max(\frac{2}{n}\sum_{i=1}^{n} I(\hat{p}_n(x_i) - 2\delta\kappa y_i, y_i), \lambda^2\delta^2/(5\kappa^2)).$$

We may obtain from Theorem 3.3 the following result: with probability of at least $1 - \eta$,

$$E_{x,y}I(\hat{p}_n(x), y) \leq \frac{1}{n}\sum_{i=1}^{n} I(\hat{p}_n(x_i) - 4\sqrt{\frac{10\ln\frac{3}{\eta}}{\lambda^2 n}}\kappa^2 y_i, y_i) + \sqrt{\frac{\ln\frac{3}{\eta}}{2n}}.$$

It is interesting to compare this result with margin percentile style bounds from VC analysis. For example, Theorem 4.19 in [2] implies that there exists a constant $C$ such that with probability of at least $1 - \eta$: for all $\gamma$ we have

$$E_{x,y}I(\hat{p}_n(x), y) \leq \frac{1}{n}\sum_{i=1}^{n} I(\hat{p}_n(x_i) - \gamma y_i, y_i) + C\sqrt{\frac{\kappa^2\ln^2 n}{\lambda\gamma^2 n} + \frac{\ln\frac{1}{\eta}}{n}}.$$

We can see that if we assume that $\lambda$ is small and the margin $\gamma = 4\sqrt{10\frac{\ln\frac{3}{\eta}}{\lambda^2 n}}\kappa^2$ is also small, then the above bound with this choice of $\gamma$ is inferior to the bound in Theorem 3.3. Clearly, this implies that our analysis has some advantages over VC analysis due to the fact that we directly analyze the numerical formulation of support vector classification.

## 4 Conclusion

In this paper, we have introduced a notion of the convergence of the estimated predictor to the best underlying predictor for some learning problems in Hilbert spaces. This generalizes an earlier study in [6]. We derived generalization bounds for some regression and classification problems. We have shown that results from our analysis compare favorably with a number of earlier studies. This indicates that the concept introduced in this paper can lead to valuable insights into certain numerical formulations of learning problems.

## Footnotes

[1]In [1], there was a small error after equation (11). As a result, the original bound in their paper was in a form equivalent to the one we cite here with $\kappa$ replaced by $\kappa^{1/2}$.

## References

[1] Olivier Bousquet and André Elisseeff. Algorithmic stability and generalization performance. In Todd K. Leen, Thomas G. Dietterich, and Volker Tresp, editors, *Advances in Neural Information Processing Systems 13*, pages 196–202. MIT Press, 2001.

[2] Nello Cristianini and John Shawe-Taylor. *An Introduction to Support Vector Machines and other Kernel-based Learning Methods*. Cambridge University Press, 2000.

[3] Wee Sun Lee, Peter L. Bartlett, and Robert C. Williamson. The importance of convexity in learning with squared loss. *IEEE Trans. Inform. Theory*, 44(5):1974–1980, 1998.

[4] R. Tyrrell Rockafellar. *Convex analysis*. Princeton University Press, Princeton, NJ, 1970.

[5] Vadim Yurinsky. *Sums and Gaussian vectors*. Springer-Verlag, Berlin, 1995.

[6] Tong Zhang. Convergence of large margin separable linear classification. In *Advances in Neural Information Processing Systems 13*, pages 357–363, 2001.

[7] Tong Zhang. A leave-one-out cross validation bound for kernel methods with applications in learning. In *14th Annual Conference on Computational Learning Theory*, pages 427–443, 2001.
